# History-dependent Attractor Neural Networks

**Isaac Meilijson**          **Eytan Ruppin**
School of Mathematical Sciences
Raymond and Beverly Sackler Faculty of Exact Sciences
Tel-Aviv University, 69978 Tel-Aviv, Israel.

## Abstract

We present a methodological framework enabling a detailed description of the performance of Hopfield-like attractor neural networks (ANN) in the first two iterations. Using the Bayesian approach, we find that performance is improved when a history-based term is included in the neuron's dynamics. A further enhancement of the network's performance is achieved by judiciously choosing the *censored* neurons (those which become active in a given iteration) on the basis of the magnitude of their post-synaptic potentials. The contribution of biologically plausible, censored, history-dependent dynamics is especially marked in conditions of low firing activity and sparse connectivity, two important characteristics of the mammalian cortex. In such networks, the performance attained is higher than the performance of two 'independent' iterations, which represents an upper bound on the performance of history-independent networks.

## 1   Introduction

Associative Attractor Neural Network (ANN) models provide a theoretical background for the understanding of human memory processes. Considerable effort has been devoted recently to narrow the gap between the original ANN Hopfield model (Hopfield 1982) and the realm of the structure and dynamics of the brain (e.g., Amit & Tsodyks 1991). In this paper, we contribute to the examination of the performance of ANNs under cortical-like architectures, where neurons are typically connected to only a fraction of their neighboring neurons, and have a low firing activity (Abeles et. al. 1990). We develop a general framework for examining various *signalling* mechanisms (firing functions) and *activation* rules (the mechanism for deciding which neurons are active in some interval of time).

The Hopfield model is based on *memoryless dynamics*, which identify the notion of 'post-synaptic potential' with the input field received by a neuron from the neurons active in the current iteration. We follow a Bayesian approach under which the neuron's signalling and activation decisions are based on the current a-posteriori probabilities assigned to its two possible true memory states, ±1. As we shall see, the a-posteriori belief in +1 is the sigmoidal function evaluated at a neuron's *generalized field*, a linear combination of present and past input fields. From a biological perspective, this *history-dependent* approach is strongly motivated by the observation that the time span of the different channel conductances in a given neuron is very broad (see Lytton 1991 for a review). While some channels are active for only microseconds, some slow-acting channels may remain open for seconds. Hence, a synaptic input currently impending on the neuron may influence both its current post-synaptic membrane potential, and its post-synaptic potential at some future time.

## 2  The Model

The neural network model presented is characterized as follows. There are $m$ 'random memories' $\xi^\mu$, $1 \le \mu \le m$, and one 'true' memory $\xi^{m+1} = \xi$. The $(m+1)N$ entries of these memories are independent and identically distributed, with equally likely values of +1 or $-1$. The initial state $X$ has *similarity* $P(X_i = \xi_i) = (1+\epsilon)/2$, $P(X_i = -\xi_i) = (1 - \epsilon)/2$, independently of everything else. The weight of the synaptic connection between neurons $i$ and $j$ $(i \ne j)$ is given by the simple Hebbian law

$$W_{ij}^* = \sum_{\mu=1}^{m+1} \xi^\mu{}_i \xi^\mu{}_j \tag{1}$$

Each neuron receives incoming synaptic connections from a random choice of $K$ of the $N$ neurons in the network in such a way that if a synapse exists, the synapse in the opposite direction exists with probability $r$, the *reflexivity* parameter. In the first iteration, a random sample of $L_1$ neurons become *active* (i.e., 'fire'), thus on the average $n_1 = L_1 K/N$ neurons update the state of each neuron. The field $f_i^{(1)}$ of neuron $i$ in the first iteration is

$$f_i^{(1)} = \frac{1}{n_1} \sum_{j=1}^N W_{ij}^* I_{ij} I_j^{(1)} X_j , \tag{2}$$

where $I_{ij}$ denotes the indicator function of the event 'neuron $i$ receives a synaptic connection from neuron $j$', and $I_j^{(t)}$ denotes the indicator function of the event 'neuron $j$ is active in the t'th iteration'. Under the Bayesian approach we adopt, neuron $i$ assigns an a-priori probability $\lambda_i^{(0)} = P(\xi_i = +1|X_i) = (1 + \epsilon X_i)/2$ to having +1 as the correct memory state and evaluates the corresponding a-posteriori probability $\lambda_i^{(1)} = P(\xi_i = +1|X_i, f_i^{(1)})$, which turns out to be expressible as the

sigmoidal function $1/(1 + exp(-2x))$ evaluated at some linear combination of $X_i$ and $f_i^{(1)}$.

In the second iteration the belief $\lambda_i^{(1)}$ of a neuron determines the probability that the neuron is active. We illustrate two extreme modes for determining the active updating neurons, or *activation*: the *random* case where $L_2$ active neurons are randomly chosen, independently of the strength of their fields, and the *censored* case, which consists of selecting the $L_2$ neurons whose belief belongs to some set. The most appealing censoring rule from the biological point of view is *tail-censoring*, where the active neurons are those with the strongest beliefs. Performance, however, is improved under *interval-censoring*, where the active neurons are those with mid-range beliefs, and even further by combining tail and interval censoring into a *hybrid* rule.

Let $n_2 = L_2 K/N$. The activation rule is given by a function $C : [\frac{1}{2}, 1] \rightarrow [0, 1]$. Neuron $j$, with belief $\lambda_j^{(1)}$ in $+1$, becomes active with probability $C(max(\lambda_j^{(1)}, 1 - \lambda_j^{(1)}))$, independently of everything else. For example, the random case corresponds to $C \equiv \frac{L_2}{N}$ and the tail-censored case corresponds to $C(\lambda) = 1$ or $0$ depending on whether $max(\lambda, 1 - \lambda)$ exceeds some threshold. The output of an active neuron $j$ is a *signal function* $S(\lambda_j^{(1)})$ of its current belief. The field $f_i^{(2)}$ of neuron $i$ in the second iteration is

$$f_i^{(2)} = \frac{1}{n_2} \sum_{j=1}^{N} W_{ij}{}^* I_{ij} I_j^{(2)} S(\lambda_j^{(1)}) . \tag{3}$$

Neuron $i$ now evaluates its a-posteriori belief $\lambda_i^{(2)} = P(\xi_i = +1 | X_i, I_i^{(1)}, f_i^{(1)}, f_i^{(2)})$. As we shall see, $\lambda_i^{(2)}$ is, again, the sigmoidal function evaluated at some linear combination of the neuron's history $X_i, X_i I_i^{(1)}, f_i^{(1)}$ and $f_i^{(2)}$. In contrast to the common history-independent Hopfield dynamics where the signal emitted by neuron $j$ in the $t'th$ iteration is a function of $f_j^{(t-1)}$ only, Bayesian history-dependent dynamics involve signals and activation rules which depend on the neuron's generalized field, obtained by adaptively incorporating $f_j^{(t-1)}$ to its previous generalized field. The final state $X_i^{(2)}$ of neuron $i$ is taken as $-1$ or $+1$, depending on which of $1 - \lambda_i^{(2)}$ and $\lambda_i^{(2)}$ exceeds $1/2$.

For $n_1/N, n_2/N, m/N, K/N$ constant, and $N$ large, we develop explicit expressions for the performance of the network, for any signal function (e.g., $S_1(\lambda) = Sgn(\lambda - 1/2)$ or $S_2(\lambda) = 2\lambda - 1$) and activation rule. Performance is measured by the final *overlap* $\epsilon'' = \frac{1}{N} \sum \xi_i X_i^{(2)}$ (or equivalently by the final similarity $(1 + \epsilon'')/2$). Various possible combinations of activation modes and signal functions described above are then examined under varying degrees of connectivity and neuronal activity.

## 3    Single-iteration optimization: the Bayesian approach

Consider the following well known basic fact in Bayesian Hypothesis Testing,
Lemma 1
Express the prior probability as

$$P(\xi = 1) = \frac{1}{1 + e^{-2x}} \tag{4}$$

and assume an observable $Y$ which, given $\xi$, is distributed according to

$$Y|\xi \sim N(\mu\xi, \sigma^2) \tag{5}$$

for some constants $\mu \in (-\infty, \infty)$ and $\sigma^2 \in (0, \infty)$. Then the posterior probability is

$$P(\xi = 1|Y = y) = \frac{1}{1 + e^{-2(x + (\mu/\sigma^2)y)}}. \tag{6}$$

Applying this Lemma to $Y = f_i^{(1)}$, with $\mu = \epsilon$ and $\sigma^2 = \frac{m}{n_1} \equiv \alpha_1$, we see that

$$\lambda_i^{(1)} = P(\xi_i = 1|X_i, f_i^{(1)}) = \frac{1}{1 + e^{-2\epsilon(\gamma(\epsilon)X_i + f_i^{(1)}/\alpha_1)}}, \tag{7}$$

where $\gamma(\epsilon) = \frac{1}{2\epsilon} \log \frac{1+\epsilon}{1-\epsilon}$. Hence, $P(\xi = 1|X_i, f_i^{(1)}) > 1/2$ if and only if $f_i^{(1)} + \alpha_1\gamma(\epsilon)X_i > 0$. The single-iteration performance is then given by the similarity

$$\frac{1+\epsilon'}{2} = P\left((f_i^{(1)} + \alpha_1\gamma(\epsilon)X_i)\xi_i > 0|\xi_i\right) = \tag{8}$$

$$\frac{1+\epsilon}{2}\Phi\left(\frac{\epsilon}{\sqrt{\alpha_1}} + \gamma(\epsilon)\sqrt{\alpha_1}\right) + \frac{1-\epsilon}{2}\Phi\left(\frac{\epsilon}{\sqrt{\alpha_1}} - \gamma(\epsilon)\sqrt{\alpha_1}\right)$$

$$\equiv Q(\epsilon, \alpha_1)$$

where $\Phi$ is the standard normal distribution function. The Hopfield dynamics, modified by redefining $W_{ii}$ as $m\gamma(\epsilon)$ (in the Neural Network terminology) is equivalent (in the Bayesian jargon) to the obvious optimal policy, under which a neuron sets for itself the sign with posterior probability above $1/2$ of being correct.

## 4  Two-iterations optimization

For mathematical convenience, we will relate signals and activation rules to normalized generalized fields rather than to beliefs. We let

$$h(x) = S(\frac{1}{1 + e^{-2cx}}) , \quad p(x) = C\left(max\left(\frac{1}{1 + e^{-2cx}}, 1 - \frac{1}{1 + e^{-2cx}}\right)\right) \tag{9}$$

for $c = \epsilon/\sqrt{\alpha_1}$. The signal function $h$ is assumed to be odd, and the activation function $p$, even.

In order to evaluate the belief $\lambda_i^{(2)}$, we need the conditional distribution of $f_i^{(2)}$ given $X_i$, $I_i^{(1)}$ and $f_i^{(1)}$, for $\xi_i = -1$ or $\xi_i = +1$. We adopt the working assumption that the pair of random variables $(f_i^{(1)}, f_i^{(2)})$ has a bivariate normal distribution given $\xi_i$, $I_i^{(1)}$ and $X_i$, with $\xi_i$, $I_i^{(1)}$ and $X_i$ affecting means but not variances or correlations. Under this working assumption, $f_i^{(2)}$ is conditionally normal given $(\xi_i, I_i^{(1)}, X_i, f_i^{(1)})$, with constant variance and a mean which we will identify. This working assumption allows us to model performance via the following well known regression model.

Lemma 2
If two random variables $U$ and $V$ with finite variances are such that $E(V|U)$ is a linear function of $U$ and $Var(V|U)$ is constant, then

$$E(V|U) = E(V) + \frac{Cov(U,V)}{Var(U)}(U - E(U)) \tag{10}$$

and

$$Var(V|U) = Var(V) - \frac{(Cov(U,V))^2}{Var(U)}. \tag{11}$$

Letting $U = f_i^{(1)}$ and $V = f_i^{(2)}$, we obtain

$$\lambda_i^{(2)} = P(\xi_i = 1 | X_i, I_i^{(1)}, f_i^{(1)}, f_i^{(2)}) = \tag{12}$$

$$\frac{1}{1 + \exp\{-2\left[\epsilon\left(f_i^{(1)}/\alpha_1 + \gamma(\epsilon)X_i\right) + \frac{\epsilon^* - a\epsilon}{\tau^2}\left(f_i^{(2)} - bX_iI_i^{(1)} - af_i^{(1)}\right)\right]\}} =$$

$$\frac{1}{1 + \exp\{-2\left[\left(\epsilon\gamma(\epsilon) - \frac{b(\epsilon^* - a\epsilon)}{\tau^2}I_i^{(1)}\right)X_i + \left(\frac{\epsilon}{\alpha_1} - \frac{a(\epsilon^* - a\epsilon)}{\tau^2}\right)f_i^{(1)} + \frac{\epsilon^* - a\epsilon}{\tau^2}f_i^{(2)}\right]\}}$$

which is the sigmoidal function evaluated at some generalized field. Expression (12) shows that the correct definition of a final state $X_i^{(2)}$, as the most likely value among $+1$ or $-1$, is

$$X_i^{(2)} = Sgn\left[\left(\epsilon\gamma(\epsilon) - \frac{b(\epsilon^* - a\epsilon)}{\tau^2}I_i^{(1)}\right)X_i + \left(\frac{\epsilon}{\alpha_1} - \frac{a(\epsilon^* - a\epsilon)}{\tau^2}\right)f_i^{(1)} + \frac{\epsilon^* - a\epsilon}{\tau^2}f_i^{(2)}\right] \tag{13}$$

and the performance is given by

$$P(X_i^{(2)} = \xi_i|\xi_i) = \frac{1+\epsilon}{2}\Phi\left(\frac{\epsilon}{\sqrt{\alpha^*}} + \gamma(\epsilon)\sqrt{\alpha^*}\right) + \frac{1-\epsilon}{2}\Phi\left(\frac{\epsilon}{\sqrt{\alpha^*}} - \gamma(\epsilon)\sqrt{\alpha^*}\right) = \tag{14}$$

$$Q(\epsilon, \alpha^*)$$

where the one-iteration performance function $Q$ is defined by (8), and

$$\alpha^* = \frac{m}{n^*} = \frac{m}{n_1 + m\left(\frac{\epsilon^*/\epsilon - a}{\tau}\right)^2}. \tag{15}$$

We see that the performance is conveniently expressed as the single-iteration optimal performance, had this iteration involved $n^*$ rather than $n_1$ sampled neurons. This formula yields a numerical and analytical tool to assess the network's performance with different signal functions, activation rules and architectures. Due to space restrictions, the identification of the various parameters used in the above formulas is not presented. However, it can be shown that in the sparse limit arrived at by fixing $\alpha_1$ and $\alpha_2$ and letting both $K/m$ and $N/K$ go to infinity, it is always better to replace an iteration by two smaller ones. This suggests that Bayesian

updating dynamics should be essentially asynchronous. We also show that the two-iterations performance $Q\left(\epsilon, \dfrac{1}{\frac{1}{\alpha_1}+\frac{1}{\alpha_2}\left(\frac{Q(\epsilon,\alpha_1)}{\epsilon}\right)^2}\right)$ is superior to the performance $Q\left(2Q(\epsilon,\alpha_1)-1,\alpha_2\right)$ of two *independent* optimal single iterations.

## 5  Heuristics on activation and signalling

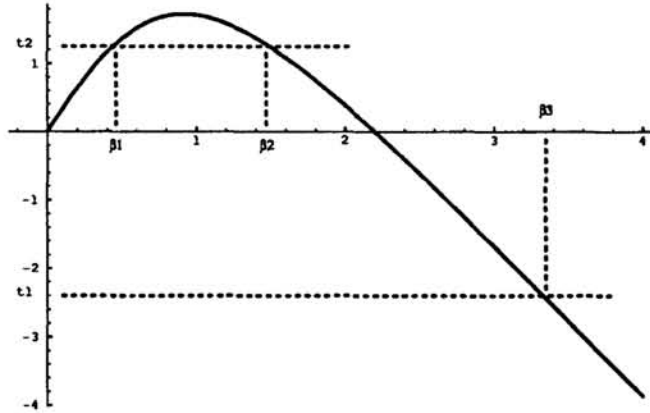

Figure 1: A typical plot of $R(x) = \phi_1(x)/\phi_0(x)$. Network parameters are $N = 500$, $K = 500$, $n_1 = n_2 = 50$ and $m = 10$.

By (14) and (15), performance is mostly determined by the magnitude of $(\epsilon^* - a\epsilon)^2$. It can be shown that

$$(\epsilon^* - a\epsilon)\bar{\Psi}_a = \int_0^\infty p(x)h(x)\phi_1(x)dx \tag{16}$$

and

$$\bar{\Psi}_a = \int_0^\infty p(x)\phi_0(x)dx \tag{17}$$

where $\phi_1$ and $\phi_0$ are some specific linear combinations of Gaussian densities and their derivatives, and $\bar{\Psi}_a = n_2/K$ is the activity level. High performance is achieved by maximizing over $p$ and possibly over $h$ the absolute value of expression (16) keeping (17) fixed. In complete analogy to Hypothesis Testing in Statistics, where $\bar{\Psi}_a$ takes the role of *level of significance* and $(\epsilon^* - a\epsilon)\bar{\Psi}_a$ the role of *power*, $p(x)$ should be 1 or 0 (activate the neuron or don't) depending on whether the field value $x$ is such that the *likelihood ratio* $h(x)\phi_1(x)/\phi_0(x)$ is above or below a given threshold, determined by (17). Omitting details, the ratio $R(x) = \phi_1(x)/\phi_0(x)$ looks as in figure 1, and converges to $-\infty$ as $x \to \infty$.

We see that there are three reasonable ways to make the ratio $h(x)\phi_1(x)/\phi_0(x)$ large: we can take a negative threshold such as $t_1$ in figure 1, activate all neurons with generalized field exceeding $\beta_3$ (tail-censoring) and signal $h(x) = -Sgn(x)$,

or take a positive threshold such as $t_2$ and activate all neurons with field value between $\beta_1$ and $\beta_2$ (interval-censoring) and signal $h(x) = Sgn(x)$. Better still, we can consider the *hybrid signalling-censoring rule*: Activate all neurons with absolute field value between $\beta_1$ and $\beta_2$, or beyond $\beta_3$. The first group should signal their preferred sign, while those in the second group should signal the sign opposite to the one they so strongly believe in !

## 6   Numerical results

| Performance | predicted | experimental |
|---|---|---|
| Random activation | 0.955 | 0.951 |
| Tail censoring | 0.972 | 0.973 |
| Interval/Hybrid censoring | 0.975 | 0.972 |
| Hopfield - zero diagonal | – | 0.902,0.973 |
| Independent $\gamma(\epsilon)$ diagonal | 0.96 | – |
| Independent zero diagonal | 0.913 | – |

Table 1: Sparsely connected, low activity network: $N = 1500, K = 50, n_1 = n_2 = 20, m = 5$.

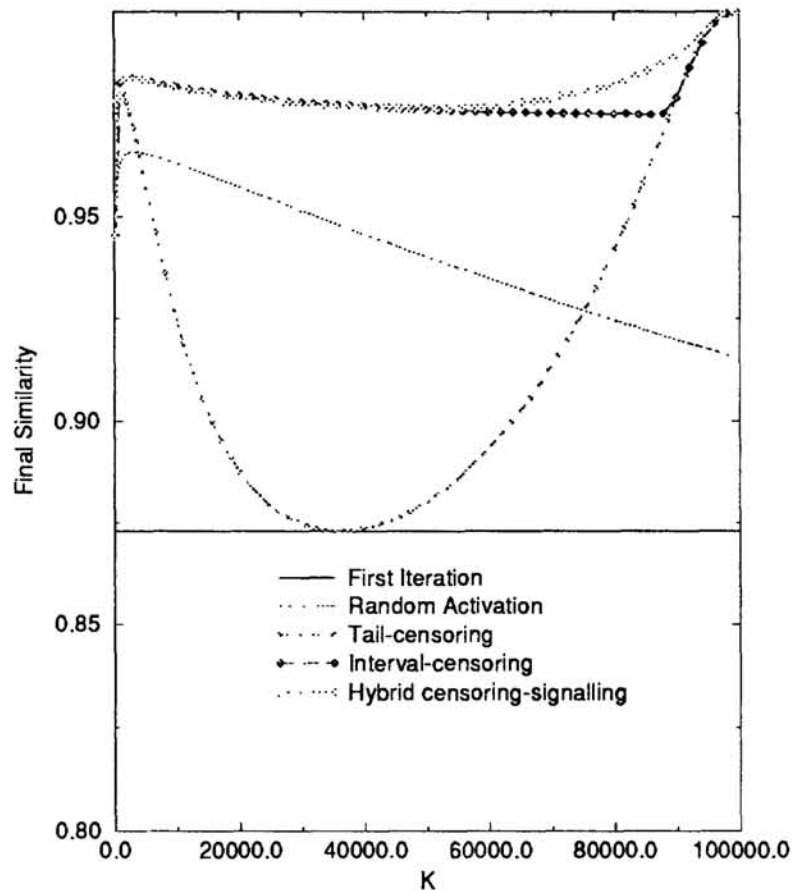

Figure 2: Performance of a large-scale cortical-like 'columnar' ANN, at different values of connectivity $K$, for initial similarity 0.75. $N = 10^5$, $n_1 = n_2 = 200$, $m = 50$. The horizontal line denotes the performance of a single iteration.

Our theoretical performance predictions show good correspondence with simulation results, already at fairly small-scale networks. The superiority of history-dependent dynamics is apparent. Table 1 shows the performance achieved in a sparsely-connected network. The predicted similarity after two iterations is reported, starting from initial similarity 0.75, and compared with experimental results averaged over 100 trials.

Figure 2 illustrates the theoretical two-iterations performance of large, low-activity 'cortical-like' networks, as a function of connectivity. We see that interval-censoring can maintain high performance throughout the connectivity range. The performance of tail-censoring is very sensitive to connectivity, almost achieving the performance of interval censoring at a narrow low-connectivity range, and becoming optimal only at very high connectivity. The superior hybrid rule improves on the others only under high connectivity. As a cortical neuron should receive the concomitant firing of about $200 - 300$ neurons in order to be activated (Treves & Rolls 1991), we have set $n = 200$. We find that the optimal connectivity per neuron, for biologically plausible tail-censoring activation, is of the same order of magnitude as actual cortical connectivity. The actual number $nN/K$ of neurons firing in every iteration is about 5000, which is in close correspondence with the evidence suggesting that about 4% of the neurons in a module fire at any given moment (Abeles et. al. 1990).

# References

[1] J.J. Hopfield. Neural networks and physical systems with emergent collective abilities. *Proc. Nat. Acad. Sci. USA*, 79:2554, 1982.

[2] D. J. Amit and M. V. Tsodyks. Quantitative study of attractor neural network retrieving at low spike rates: I. substrate–spikes, rates and neuronal gain. *Network*, 2:259–273, 1991.

[3] M. Abeles, E. Vaadia, and H. Bergman. Firing patterns of single units in the prefrontal cortex and neural network models. *Network*, 1:13–25, 1990.

[4] W. Lytton. Simulations of cortical pyramidal neurons synchronized by inhibitory interneurons. *J. Neurophysiol.*, 66(3):1059–1079, 1991.

[5] A. Treves and E. T. Rolls. What determines the capacity of autoassociative memories in the brain? *Network*, 2:371–397, 1991.
